# Bayesian Modeling and Classification of Neural Signals

**Michael S. Lewicki**

Computation and Neural Systems Program
California Institute of Technology 216–76
Pasadena, CA 91125

`lewicki@cns.caltech.edu`

## Abstract

Signal processing and classification algorithms often have limited applicability resulting from an inaccurate model of the signal's underlying structure. We present here an efficient, Bayesian algorithm for modeling a signal composed of the superposition of brief, Poisson-distributed functions. This methodology is applied to the specific problem of modeling and classifying extracellular neural waveforms which are composed of a superposition of an unknown number of action potentials (APs). Previous approaches have had limited success due largely to the problems of determining the spike shapes, deciding how many are shapes distinct, and decomposing overlapping APs. A Bayesian solution to each of these problems is obtained by inferring a probabilistic model of the waveform. This approach quantifies the uncertainty of the form and number of the inferred AP shapes and is used to obtain an efficient method for decomposing complex overlaps. This algorithm can extract many times more information than previous methods and facilitates the extracellular investigation of neuronal classes and of interactions within neuronal circuits.

# 1   INTRODUCTION

Extracellular electrodes typically record the activity of several neurons in the vicinity of the electrode tip (figure 1). Most electrophysiological data is collected by isolating action potentials (APs) from a single neuron by using a level detector or window discriminator. Methods for extracting APs from multiple neurons can, in addition to the obvious advantage of providing more data, provide the means to investigate local neuronal interactions and response properties of neuronal populations. Determining from the voltage waveform what cell fired when is a difficult, ill-posed problem which is compounded by the fact that cells frequently fire simultaneously resulting in large variations in the observed shapes.

There are three major difficulties in identifying and classifying action potentials (APs) in a neuron waveform. The first is determining the AP shapes, the second is deciding the number of distinct shapes, and the third is decomposing overlapping spikes into their component parts. In general, these problems cannot be solved independently, since the solution of one will affect the solution of the others.

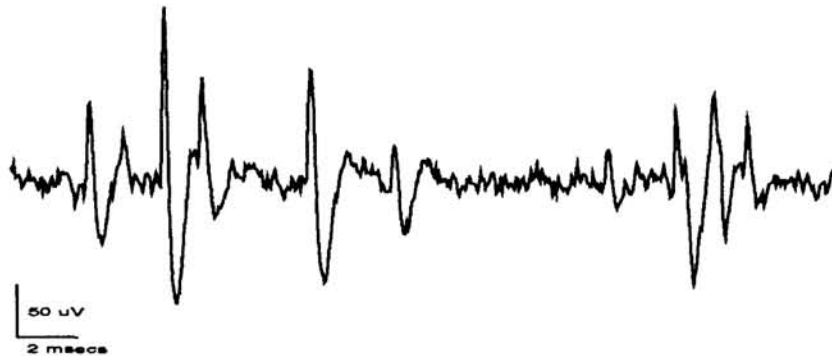

Figure 1: Each neuron generates a stereotyped action potential (AP) which is observed through the electrode as a voltage fluctuation. This shape is primarily a function of the position of a neuron relative to the tip. The extracellular waveform shows several different APs generated by an unknown number of neurons. Note the frequent presence of overlapping APs which can completely obscure individual spikes.

The approach summarized here is to model the waveform directly to obtain a probabilistic description of each action potential and, in turn, of the whole waveform. This method allows us to compute the class conditional probabilities of each AP. In addition, it is possible to quantify the certainty of both the form and number of spike shapes. Finally, we can use this description to decompose overlapping APs efficiently and assign probabilities to alternative spike sequences.

# 2   MODELING SINGLE ACTION POTENTIALS

The data from the event observed (at time zero) is modeled as resulting from a fixed underlying spike function, $s(t)$, plus noise:

$$d_i = s(t_i\ ;\mathbf{v}) + \eta_i, \tag{1}$$

where $\mathbf{v}$ is the parameter vector that defines the spike function. The noise, $\eta$, is modeled as Gaussian with zero mean and standard deviation $\sigma_\eta$.

From the Bayesian perspective, the task is to infer the posterior distribution of the spike function parameters (assuming, for the moment, that $\sigma_\eta$ and $\sigma_w$ are known):

$$P(\mathbf{v}|D, \sigma_\eta, \sigma_w, M) = \frac{P(D|\mathbf{v}, \sigma_\eta, M)\, P(\mathbf{v}|\sigma_w, M)}{P(D|\sigma_\eta, \sigma_w, M)}. \tag{2}$$

The two terms specifying the posterior distribution of $\mathbf{v}$ are 1) the probability of the data given the model:

$$P(D|\mathbf{v}, \sigma_\eta, M) = \frac{1}{Z_D(\sigma_\eta)} \exp\left[-\frac{1}{2\sigma_\eta^2} \sum_{i=1}^{I} (d_i - s(t_i))^2\right], \tag{3}$$

and 2) the prior assumptions of the structure of $s(t)$ which are assumed to be of the form:

$$P(\mathbf{v}|\sigma_w, M) \propto \exp\left[-\int du\, s^{(m)}(u)^2/\sigma_w^2\right]. \tag{4}$$

The superscript $(m)$ denotes differentiation which for these demonstrations we assumed to be $m = 1$ corresponding to linear splines. The smoothness of $s(t)$ is controlled through $\sigma_w$ with small values of $\sigma_w$ penalizing large fluctuations.

The final step in determining the posterior distribution is to eliminate the dependence of $P(\mathbf{v}|D, \sigma_\eta, \sigma_w, M)$ on $\sigma_\eta$ and $\sigma_w$. Here, we use the approximation:

$$P(\mathbf{v}|D, M) \approx P(\mathbf{v}|D, \sigma_w^{\mathrm{MP}}, \sigma_\eta^{\mathrm{MP}}, M). \tag{5}$$

The most probable values of $\sigma_\eta$ and $\sigma_w$ were obtained using the methods of MacKay (1992) in which reestimation formulas are obtained from a Gaussian approximation of the posterior distribution for $\sigma_\eta$ and $\sigma_w$, $P(\sigma_\eta, \sigma_w|D, M)$. Correct inference of $\sigma_w$ prevents the spike function from overfitting the data.

## 3   MODELING MULTIPLE ACTION POTENTIALS

When a waveform contains multiple types of APs, determining the component spike shapes is more difficult because the classes are not known *a priori*. The uncertainty of which class an event belongs to can be incorporated with a *mixture distribution*. The probability of a particular event, $\mathbf{D}_n$, given all spike models, $M_{1:\mathrm{K}}$, is

$$P(\mathbf{D}_n|\mathbf{v}_{1:\mathrm{K}}, \boldsymbol{\pi}, \sigma_\eta, M_{1:\mathrm{K}}) = \sum_{k=1}^{K} \pi_k P(\mathbf{D}_n|\mathbf{v}_k, \sigma_\eta, M_k), \tag{6}$$

where $\pi_k$ is the *a priori* probability that a spike will be an instance of $M_k$, and $\sum \pi_k = 1$.

As before, the objective is to determine the posterior distribution for the parameters defining a set of spike models, $P(\mathbf{v}_{1:\mathrm{K}}, \boldsymbol{\pi}|\mathbf{D}_{1:\mathrm{N}}, \sigma_\eta, \sigma_w, M_{1:\mathrm{K}})$ which is obtained again using Bayes' rule.

Finding the conditions satisfied at a posterior maximum leads to the equation:

$$\sum_{n=1}^{N} P(M_k|\mathbf{D}_n, \mathbf{v}_k, \boldsymbol{\pi}, \sigma_\eta)\frac{1}{\sigma_\eta^2}\sum_i [d_{n,i} - s_k(t_i - \tau_n ; \mathbf{v}_k)]\frac{\partial s_k(t_i ; \mathbf{v}_k)}{\partial \mathbf{v}_k} = 0, \quad (7)$$

where $\tau_n$ is the inferred occurrence time (typically to sub-sample period accuracy) of the event $\mathbf{D}_n$. This equation is solved iteratively to obtain the most probable values of $\mathbf{v}_{1:K}$. Note that the error for each event, $\mathbf{D}_n$, is weighted by $P(M_k|\mathbf{D}_n, \mathbf{v}_k, \boldsymbol{\pi}, \sigma_\eta)$ which is the probability that the event is an instance of the $k$th spike model. This is a *soft* clustering procedure, since the events are not explicitly assigned to particular classes. Maximizing the posterior yields accurate estimates of the spike functions even when the clusters are highly overlapping.

The techniques described in the previous section are used to determine the most probable values for $\sigma_\eta$ and $\sigma_w$ and, in turn, the most probable values of $\mathbf{v}_{1:K}$ and $\boldsymbol{\pi}$.

## 4   DETERMINING THE NUMBER OF SPIKE SHAPES

Choosing a set of spike models that best fit the data, would result eventually in a model for each event in the waveform. Heuristics might indicate whether two spike models are identical or distinct, but *ad hoc* criteria are notoriously dependent on particular circumstances, and it is difficult to state precisely what information the rules take into account.

To determine the *most probable* number of spike models, we apply probability theory. Let $S_j = \{M_{1:K}^{(j)}\}$ denote a set of spike models and $H$ denote information known *a priori*. The probability of $S_j$, conditioned only on $H$ and the data, is obtained using Bayes' rule:

$$P(S_j|\mathbf{D}_{1:N}, H) = \frac{P(S_j|H)P(\mathbf{D}_{1:N}|S_j, H)}{P(\mathbf{D}_{1:N}|H)}. \quad (8)$$

The only data-dependent term is $P(\mathbf{D}_{1:N}|S_j, H)$ which is the *evidence* for $S_j$ (MacKay, 1992). With the assumption that all hypotheses $S_{1:J}$ are equally probable *a priori*, $P(\mathbf{D}_{1:N}|S_j, H)$ ranks alternative spike sets in terms of their *probability*.

The evidence term $P(\mathbf{D}_{1:N}|S_j, H)$ is convenient because it is the normalizing constant for the posterior distribution of the parameters defining the spike set. Although calculation of $P(\mathbf{D}_{1:N}|S_j, H)$ is analytically intractable, it is often well-approximated with a Gaussian integral which was the approximation used for these demonstrations.

A convenient way of collapsing the spike set is to compare spike models pairwise. Two models in the spike set are selected along with a sampled set of events fit by each model. We then evaluate $P(\mathbf{D}|S_1)$ and $P(\mathbf{D}|S_2)$. $S_1$ is the hypothesis that the data is modeled by a single spike shape, $S_2$ says there are two spike shapes. If $P(\mathbf{D}|S_1) > P(\mathbf{D}|S_2)$, we replace both models in $S_2$ by the one in $S_1$. The procedure terminates when no more pairs can be combined to increase the evidence.

## 5   DECOMPOSING OVERLAPPING SPIKES

Overlaps must be decomposed into their component spikes for accurate inference of the spike functions and accurate classification of the events. Determining the best-fitting decomposition is difficult because of the enormous number of possible spike sequences, not only all possible model combinations for each event but also all possible event times.

A brute-force approach to this problem is to perform an exhaustive search of the space of overlapping spike functions and event times to find the sequence with maximum probability. This approach was used by Atiya (1992) in the case of two overlapping spikes with the times optimized to one sample period. Unfortunately, this is often computationally too demanding even for off-line analysis.

We make this search efficient utilizing dynamic programming and k-dimensional trees (Friedman *et al.*, 1977). Once the best-fitting decomposition can be obtained, however, it may not be optimal, since adding more spike shapes can overfit the data. This problem is minimized by evaluating the probability for alternative decompositions to determine the *most probable* spike sequence (figure 2).

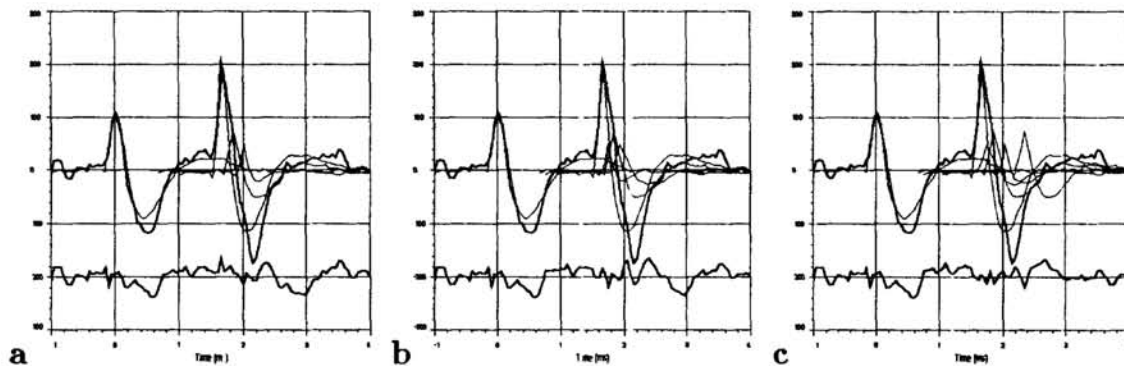

Figure 2: Many spike function sequences can account for the same region of data. The thick lines show the data, thin lines show individual spike functions. In this case, the best-fitting overlap solution is not the most probable: the sequence with 4 spike functions is more than 8 times more probable than the other solutions, even though these have smaller mean squared error. Using the *best-fitting* overlap solution may *increase* the classification error. Classification error is *minimized* by using the overlap solution that is *most probable*.

## 6   PERFORMANCE

The algorithm was tested on 40 seconds of neurophysiological data. The task is to determine the form and number of spike shapes in a waveform and to infer the occurrence times of each spike shape. The output of the algorithm is shown in figure 3. The uniformity of the residual error indicates that the six inferred spike shapes account for the entire 40 seconds of data. The spike functions $M_2$ and $M_3$ appear similar by eye, but the probabilities calculated with the methods in section 4 indicate that the two functions are significantly different. When plotted against each

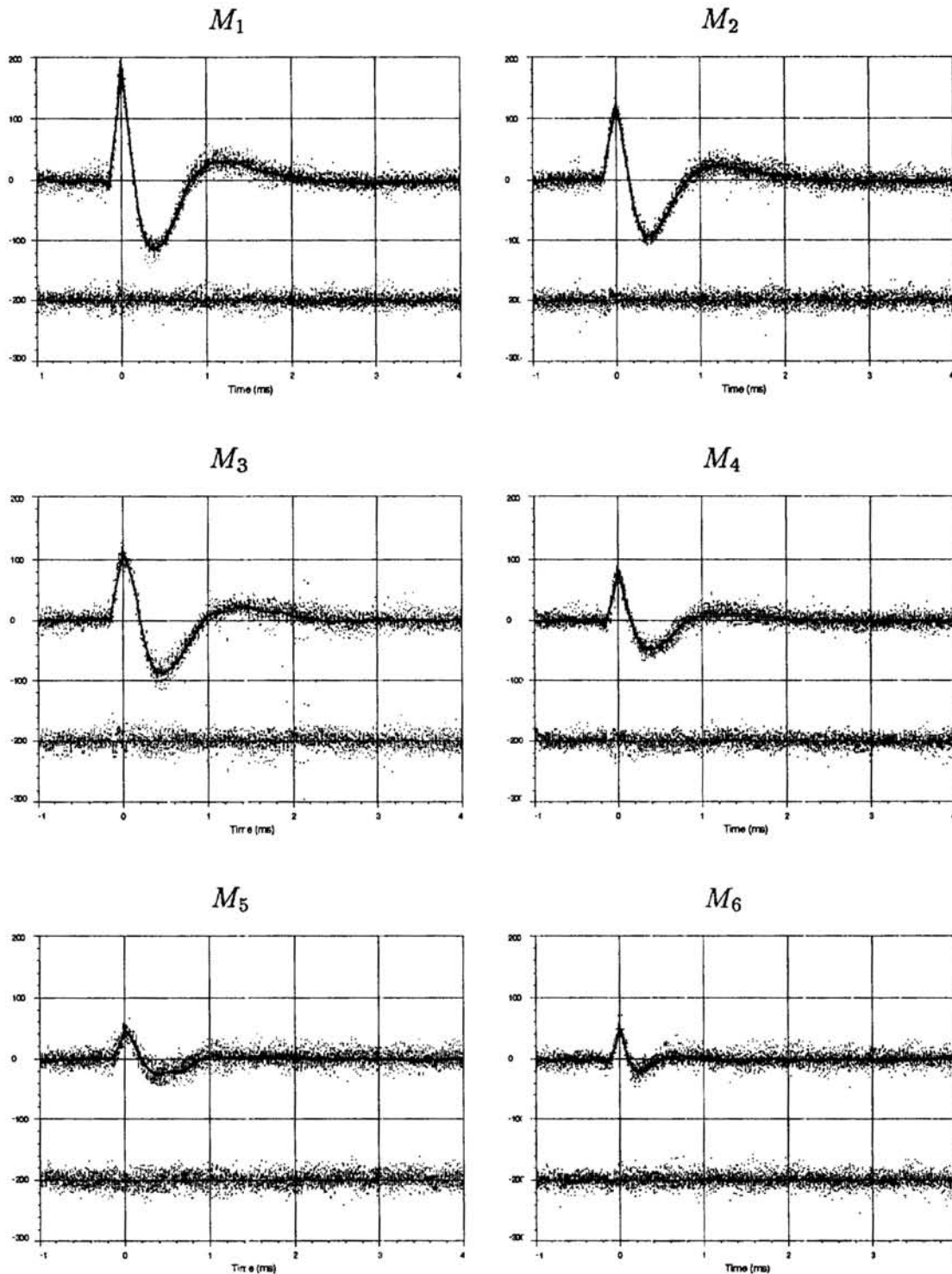

Figure 3: The solid lines are the inferred spike models. The data overlying each model is a sample of at most 40 events with overlapping spikes subtracted out. The residual errors are plotted below each model. This spike set was obtained after three iterations of the algorithm, decomposing overlaps and determining the most probable number of spike functions after each iteration. The whole inference procedure used 3 minutes of CPU time on a Sparc IPX. Once the spike set is inferred, classification of the same 40 second waveform takes about 10 seconds.

other, the two populations of APs are distinctly separated in the region around the peak with $M_3$ being wider than $M_2$.

The accuracy of the algorithm was tested by generating an artificial data set composed of the six inferred shapes shown in figure 3. The event times were Poisson distributed with frequency equal the inferred firing rate of the real data set. Gaussian noise was then added with standard deviation equal to $\sigma_\eta$. The classification results are summarized in the tables below.

Table 1: Results of the spike model inference algorithm on the synthesized data set.

| Model | 1 | 2 | 3 | 4 | 5 | 6 |
|---|---|---|---|---|---|---|
| $\Delta_{\max}/\sigma_\eta$ | 0.44 | 0.36 | 1.07 | 0.78 | 0.84 | 0.40 |

The number of spike models was correctly determined by the algorithm with the six-model spike set was preferred over the most probable five-model spike set by $\exp(34):1$ and over the most probable seven-model spike set by $\exp(19):1$. The inferred shapes were accurate to within a maximum error of $1.07\sigma_\eta$. The row elements show the maximum absolute difference, normalized by $\sigma_\eta$, between each true spike function and the corresponding inferred function.

Table 2: Classification results for the synthesized data set (non-overlapping events).

| True Models | Inferred Models | | | | | | Missed Events | Total Events |
|---|---|---|---|---|---|---|---|---|
| | 1 | 2 | 3 | 4 | 5 | 6 | | |
| 1 | 17 | 0 | 0 | 0 | 0 | 0 | 0 | 17 |
| 2 | 0 | 25 | 1 | 0 | 0 | 0 | 0 | 26 |
| 3 | 0 | 0 | 15 | 0 | 0 | 0 | 0 | 15 |
| 4 | 0 | 0 | 0 | 116 | 0 | 0 | 1 | 117 |
| 5 | 0 | 0 | 0 | 0 | 56 | 0 | 17 | 73 |
| 6 | 0 | 0 | 0 | 0 | 0 | 393 | 254 | 647 |

Table 3: Classification results for the synthesized data set (overlapping events).

| True Models | Inferred Models | | | | | | Missed Events | Total Events |
|---|---|---|---|---|---|---|---|---|
| | 1 | 2 | 3 | 4 | 5 | 6 | | |
| 1 | 22 | 0 | 0 | 0 | 0 | 0 | 0 | 22 |
| 2 | 0 | 36 | 1 | 0 | 0 | 0 | 0 | 37 |
| 3 | 0 | 0 | 20 | 0 | 0 | 0 | 0 | 20 |
| 4 | 0 | 1 | 0 | 116 | 0 | 1 | 3 | 121 |
| 5 | 0 | 0 | 0 | 1 | 61 | 1 | 19 | 82 |
| 6 | 0 | 0 | 0 | 3 | 2 | 243 | 160 | 408 |

Tables 2 and 3: Each matrix component indicates the number of times true model $i$ was classified as inferred model $j$. Events were missed if the true spikes were not detected in an overlap sequence or if all sample values for the spike fell below the event detection threshold ($4\sigma_\eta$). There was 1 false positive for $M_5$ and 7 for $M_6$.

# 7   DISCUSSION

Formulating the task as having to infer a probabilistic model made clear what was necessary to obtain accurate spike models. The soft clustering procedure accurately determines the spike shapes even when the true underlying shapes are similar. Unless the spike shapes are well-separated, commonly used hard clustering procedures will lead to inaccurate estimates.

Probability theory also allowed for an objective means of determining the number of spike models which is an essential reason for the success of this algorithm. With the wrong number of spike models overlap decomposition becomes especially difficult. The evidence has proved to be a sensitive indicator of when two classes are distinct.

Probability theory is also essential to accurate overlap decomposition. Simply fitting data with compositions of spike models leads to the same overfitting problem encountered in determining the number of spike models and in determining the spike shapes. Previous approaches have been able to handle only a limited class of overlaps, mainly due to the difficultly in making the fit efficient. The algorithm used here can fit an overlap sequence of virtually arbitrary complexity in milliseconds.

In practice, the algorithm extracts many times more information from a neural waveform than previous methods. Moreover, this information is qualitatively different from a simple list of spike times. Having reliable estimates of the action potential shapes makes it possible to study the properties of these classes, since distinct neuronal types can have distinct neuronal spikes. Finally, accurate overlap decomposition makes it possible to investigate interactions among local neurons which were previously very difficult to observe.

### Acknowledgements

I thank David MacKay for helpful discussions and Jamie Mazer for many conversations and extensive help with the development of the software. This work was supported by Caltech fellowships and an NIH Research Training Grant.

### References

A.F. Atiya. (1992) Recognition of multiunit neural signals. *IEEE Transactions on Biomedical Engineering* **39**(7):723–729.

J.H. Friedman, J.L. Bently, and R.A. Finkel. (1977) An algorithm for finding best matches in logarithmic expected time. *ACM Trans. Math. Software* **3**(3):209–226.

D. J. C. MacKay. (1992) Bayesian interpolation. *Neural Computation* **4**(3):415–445.